# Sodium entry efficiency during action potentials: A novel single-parameter family of Hodgkin-Huxley models

**Anand Singh**
Institute of Pharmacology and Toxicology
University of Zürich, Zürich, Switzerland
anands@pharma.uzh.ch

**Renaud Jolivet**[*]
Institute of Pharmacology and Toxicology
University of Zürich, Zürich, Switzerland
renaud.jolivet@a3.epfl.ch

**Pierre J. Magistretti**[†]
Brain Mind Institute
EPFL, Lausanne, Switzerland
pierre.magistretti@epfl.ch

**Bruno Weber**
Institute of Pharmacology and Toxicology
University of Zürich, Zürich, Switzerland
bweber@pharma.uzh.ch

## Abstract

Sodium entry during an action potential determines the energy efficiency of a neuron. The classic Hodgkin-Huxley model of action potential generation is notoriously inefficient in that regard with about 4 times more charges flowing through the membrane than the theoretical minimum required to achieve the observed depolarization. Yet, recent experimental results show that mammalian neurons are close to the optimal metabolic efficiency and that the dynamics of their voltage-gated channels is significantly different than the one exhibited by the classic Hodgkin-Huxley model during the action potential. Nevertheless, the original Hodgkin-Huxley model is still widely used and rarely to model the squid giant axon from which it was extracted. Here, we introduce a novel family of Hodgkin-Huxley models that correctly account for sodium entry, action potential width and whose voltage-gated channels display a dynamics very similar to the most recent experimental observations in mammalian neurons. We speak here about a family of models because the model is parameterized by a unique parameter the variations of which allow to reproduce the entire range of experimental observations from cortical pyramidal neurons to Purkinje cells, yielding a very economical framework to model a wide range of different central neurons. The present paper demonstrates the performances and discuss the properties of this new family of models.

## 1 Introduction

Action potentials play the central role in neuron-to-neuron communication. At the onset of an action potential, the change in the membrane potential leads to opening of voltage-gated sodium channels, leading to influx of sodium ions. Once the membrane is sufficiently depolarized, the opening of voltage-gated potassium channels leads to an efflux of potassium ions and brings the membrane back to the resting potential. During and after this process, the ionic gradients are restored by the Na,K-ATPase electrogenic pump which extrudes 3 sodium ions in exchange for 2 potassium ions and requires 1 ATP molecule per cycle.

---

[*]Contact author.

[†]Second affiliation: Center for Psychiatric Neuroscience, University of Lausanne, Lausanne, Switzerland.

There is thus a metabolic cost in terms of ATP molecules to be spent associated with every action potential. This metabolic cost can be roughly estimated to be 1/3 of the sodium entry into the neuron. A metabolically efficient action potential would have sodium entry restricted to the rising phase of the action potential so that a minimal number of charges is transported to produce the observed voltage change. This can be encapsulated into a measure called Sodium Entry Ratio (SER) defined as the integral of the sodium current during the action potential divided by the product of the membrane capacitance by the observed change in membrane voltage. A metabolically optimally efficient neuron would have a SER of 1 or close to 1.

The metabolic efficiency critically depends on the gating kinetics of the voltage-dependent channels and on their interaction during the action potential. All biophysical models of action potential generation rely on the framework originally established by Hodgkin and Huxley [1] and certain models in use today still rely on their parameters for the voltage-gated sodium and potassium channels responsible for the action potential generation, even though parameterization of the Hodgkin-Huxley model optimized for certain mammalian neurons have been available and used for years [2,3]. Analyzing the squid giant axon action potential, Hodgkin and Huxley established that the SER is approximately 4, owing to the fact that the sodium channels remain open during the falling phase of the action potential [1]. This has led to the idea that action potentials are metabolically inefficient and these numbers were used as key input in a number of studies aiming at establishing an energy budget for brain tissue (see e.g. [4]). However, two recent studies have demonstrated that mammalian neurons, having fundamentally similar action potentials as the squid giant axon, are significantly more efficient owing to lesser sodium entry during the falling phase of the action potential [5,6].

In the first study, Alle and colleagues observed that action potentials in mossy fiber boutons of hippocampal granule neurons have about 30% extra sodium entry than the theoretical minimum [5] (SER $\simeq 1.3$). In the second study, Carter and Bean expanded this finding, showing that different central neurons have different SERs [6]. More specifically, they measured that cortical pyramidal neurons are the most efficient with a SER $\simeq 1.2$ while pyramidal neurons from the CA1 hippocampus region have a SER $\simeq 1.6$. On the other hand, inhibitory neurons were found to have less efficient action potentials with cerebellar Purkinje neurons having a SER $\simeq 2$ and cortical basket cell interneurons having a SER $\simeq 2$. Interestingly, this is postulated to originate in the type or distribution of voltage-gated potassium channels present in each of these cell types. Even the less efficient neurons are twice more metabolically efficient than the original Hodgkin-Huxley neuron. These recent findings call for a revision of the original Hodgkin-Huxley model which fails on several accounts to describe accurately central mammalian neurons.

The aim of the present work is to formulate an *in silico* model for an accurate description of the sodium and potassium currents underlying the generation of action potentials in central mammalian neurons. To this end, we introduce a novel family of Hodgkin-Huxley models $HH_\xi$ parameterized by a single parameter $\xi$. Varying $\xi$ in a meaningful range allows reproducing the whole range of observations of Carter and Bean [6] providing a very economic modeling strategy that can be used to model a wide range of central neurons from cortical pyramidal neurons to Purkinje cells.

The next section provides a brief description of the model, of the strategy to design it as well as a formal definition of the key parameters like the Sodium Entry Ratio against which the predictions of our family of models is compared. The third section demonstrates the performances of the novel family of models and characterize its properties. Finally the last section discusses the implications of our results.

## 2 Model and methods

### 2.1 Hodgkin-Huxley model family

In order to develop a novel family of Hodgkin-Huxley models, we started from the original Hodgkin-Huxley formalism [1]. In this formalism, the evolution of the membrane voltage $V$ is governed by

$$C \frac{dV}{dt} = -\sum_k I_k + I_{\text{ext}} \qquad (1)$$

with $C$ the membrane capacitance and $I_{\text{ext}}$ an externally applied current. The currents $I_k$ are transmembrane ionic currents. Following the credo, they are described by

$$-\sum_k I_k = g_{\text{Na}}\, m^3\, h\, (V - E_{\text{Na}}) + g_{\text{K}}\, n^4\, (V - E_{\text{K}}) + g_L\, (V - E_L) \tag{2}$$

with $g_{\text{Na}}$, $g_{\text{K}}$ and $g_L$ the ionic conductances and $E_{\text{Na}}$, $E_{\text{K}}$ and $E_L$ the reversal potentials associated with the sodium current iNa $= g_{\text{Na}}\, m^3\, h\, (V - E_{\text{Na}})$, the potassium current iK $= g_{\text{K}}\, n^4\, (V - E_{\text{K}})$ and the uncharacterized leak current iL $= g_L\, (V - E_L)$. All three gating variables $m$, $n$ and $h$ follow the generic equation

$$\frac{dx}{dt} = \alpha_x(V)\,(1 - x) - \beta_x(V)\, x \tag{3}$$

with $x$ standing alternatively for $m$, $n$ or $h$. The terms $\alpha_x$ and $\beta_x$ are non-trivial functions of the voltage $V$. It is sometimes useful to reformulate Eq. 3 as

$$\frac{dx}{dt} = -\frac{1}{\tau_x(V)}\,(x - x_\infty(V)) \tag{4}$$

in which the equilibrium value $x_\infty = \alpha_x/(\alpha_x + \beta_x)$ is reached with the time constant $\tau_x = 1/(\alpha_x + \beta_x)$ which has units of [ms].

Specific values for the constants ($C$, $g_x$ and $E_x$) and for the functions $\alpha_x$ and $\beta_x$ were originally chosen to match those introduced in [7] with the exception that the model introduced in [7] includes a secondary potassium channel that was abandoned here, thus retaining only the channels originally described by Hodgkin and Huxley. The reversal potentials $E_x$ were then adjusted to match known concentrations of the respective ions in and around mammalian cells.

We then proceeded to explore the behavior of the model and observed that the specific dynamics of iNa and iK during an action potential is critically dependent on the exact definition of $\alpha_n$. In our case, $\alpha_n$ is defined by

$$\alpha_n(V) = \frac{p_1\, V - p_2}{1 - e^{-(p_3\, V - p_4)/p_5}} \tag{5}$$

with $p_1$, ..., $p_5$ some parameters. More specifically, we observed that by varying $p_5$ in a meaningful range, we could reproduce qualitatively the observations of Carter and Bean [6] regarding the dynamics of the sodium current iNa during individual action potentials.

Building on these premises, we set $p_5 = \xi$ with $\xi$ varying in the range $10.5 \leq \xi \leq 16$. These boundary values were chosen relatively arbitrarily by exploring the range in which the models stay close to experimental observations. All the other parameters appearing in the $\alpha_x$ and $\beta_x$ functions were then optimized using a standard optimization algorithm so that the model reproduces as closely as possible the values characterizing action potential dynamics as reported in [6].

The final values for parameters of the novel family of Hodgkin-Huxley models are reported in Table 1. The values of other parameters used in the model are: $C = 1.0\,\mu\text{F/cm}^2$, $g_L = 0.25$ mS/cm$^2$, $E_L = -70$ mV.

Table 1: The novel family of Hodgkin-Huxley models HH$_\xi$

| channel | variable | $\alpha_x$ | $\beta_x$ | $g_x$ (mS/cm$^2$) | $E_x$ (mV) |
|---|---|---|---|---|---|
| Na | $m$ | $\frac{41.3\,V - 3051}{1 - \exp(-\frac{V - 77.46}{13.27})}$ | $\frac{1.2499}{\exp(V/42.129)}$ | 112.7 | 50 |
| | $h$ | $\frac{0.0036}{\exp(\frac{V}{24.965})}$ | $\frac{10.405}{\exp(-\frac{1.024\,V - 26.181}{15.488}) + 1}$ | | |
| K | $n$ | $\frac{0.992\,V - 96.73}{1 - \exp(-\frac{1.042\,V - 97.517}{\xi})}$ | $\frac{0.0159}{\exp(V/21.964)}$ | 224.6 | -85.0 |

The voltage $V$ is expressed in mV.

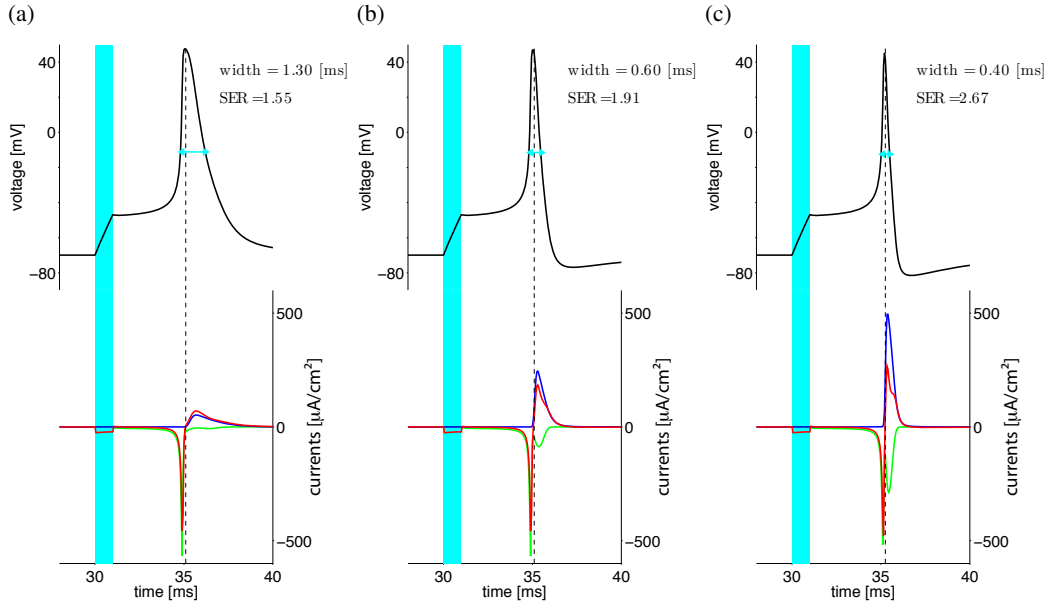

Figure 1: Dynamics of the membrane voltage $V$ (top; black line), of the sodium current iNa (bottom; green line), of the potassium current iK (bottom; blue line) and of the total current $C\,dV/dt$ (bottom; red line; see Eqs. 1-2) upon stimulation by a superthreshold pulse of current (cyan area; $I_{\text{ext}} = 25.5\,\mu A/cm^2$ for 1 ms). In each panel, SER stands for Sodium Entry Ratio (see Eq. 6) and "width" indicates the width of the action potential measured at the position indicated by the cyan arrow (see "Sodium entry ratio and numerics" subsection). (a) $\xi = 10.5$. (b) $\xi = 13.5$. (c) $\xi = 16.0$.

## 2.2 Sodium entry ratio and numerics

The relevant parameters to compare the novel family of Hodgkin-Huxley models HH$_\xi$ to the experimental dataset under consideration are: (i) the action potential peak, (ii) the action potential width and (iii) the sodium entry ratio (SER). The action potential peak is simply defined as the maximal depolarization reached during the action potential. Following [6], the action potential width is measured at half the action potential height, measured as the difference in membrane potential from the peak to the resting potential. Finally, still following [6], the SER is defined for an isolated action potential by

$$\text{SER} = \int \text{iNa}/C\Delta V \tag{6}$$

with $\Delta V$ the change in voltage during the action potential measured from the action potential threshold $\vartheta$ to its peak. The action potential threshold $\vartheta$ was defined as $1\%$ of the maximal $dV/dt$.

All simulations were implemented in MATLAB (The Mathworks, Natick MA). The system of equations was integrated using a solver for stiff problems and a time step of 0.05 ms.

## 3 Results

Recent experimental results suggest that the dynamics of the action potential generating voltage-gated channels in the classical Hodgkin-Huxley model do not correctly reproduce what is observed in mammalian neurons [5,6]. More specifically, the Hodgkin-Huxley equations generate a sodium current with a characteristic secondary peak during the action potential decaying phase, leading to a very important influx of sodium ions that counter the effect of potassium ions making the model metabolically inefficient [1]. Mammalian neurons display a sodium current with a unique sharp peak or at most a low amplitude secondary peak [5,6].

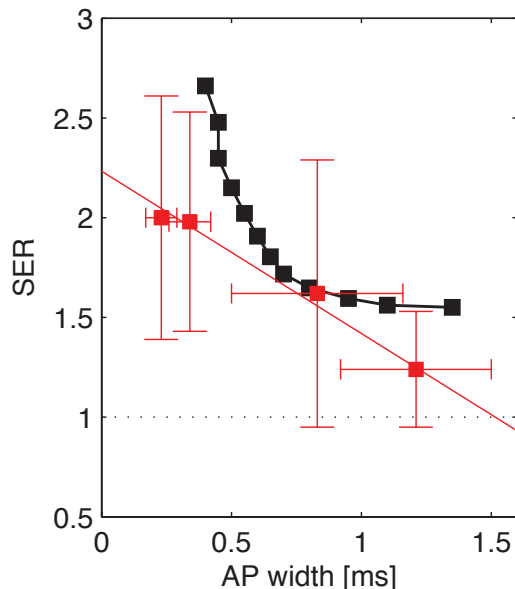

Figure 2: Predictions of our model family are compared to the experimentally observed correlation between the action potential width and the SER. Experimental observations (red squares) are adapted from [4]. Data were collected for (from left to right): Purkinje cells, cortical interneurons, CA1 pyramidal neurons and cortical pyramidal neurons. Error bars stand for the standard deviation. The red line is a simple linear regression through the experimental data ($R^2 = 0.99$). The predictions of our model (black squares) are indicated for decreasing values of $\xi$ from left ($\xi = 16$) to right ($\xi = 10.5$).

In the precedent section, we have introduced a novel family of models HH$_\xi$ parameterized by the unique parameter $\xi$ (see Table 1). We will now show how varying $\xi$ allows reproducing the wide range of dynamics observed experimentally. Figure 1 shows the behavior of HH$_\xi$ during an isolated action potential for three different values of $\xi$. In all three cases, the action potential is triggered by the same unique square pulse of current generating an isolated action potential with roughly the same latency about 4 s after the end of the stimulating pulse. Yet the behavior of the model is very different in each case. For low values of $\xi$, the sodium current iNa exhibits a single very sharp peak, being almost null after the action potential has peaked. At high values of $\xi$, iNa exhibits a distinctive secondary peak after the action potential has peaked. The potassium current iK is also much bigger in the latter case. As a consequence, the model has a low Sodium Entry Ratio (SER) at low values of $\xi$ and a high SER at high values of $\xi$ (see Eq. 6). We also observe a negative correlation between $\xi$ and the width of the action potential. The width of action potentials decreases when $\xi$ increases. Finally action potentials generated at low $\xi$ values return to the resting potential from above while action potentials generated at high $\xi$ values exhibit an after-hyperpolarization. These different instances of our family of models HH$_\xi$ cover all the experimentally observed behaviors as reported in [6] (compare with Figures 1-3 therein).

Indeed, Carter and Bean observed neurons with low SER, broad action potentials and a single sharp peak in the sodium current dynamics (cortical and CA1 pyramidal neurons). They also observed neurons with high SER, narrow action potentials and a distinctive secondary peak in the sodium current dynamics during the action potential decaying phase (cortical interneurons and cerebellar Purkinje cells). Figure 2 compares the predictions of our model family with the observations reported in [6]. It clearly demonstrates that by varying $\xi$, our model family is able to capture the whole range of observed behaviors and quantitatively fits the measured SER and action potential widths. We also observe a faint positive correlation between the action potential width and its peak like in [6] (not shown).

While the dynamics of gating variables is traditionally formulated in terms of $\alpha_x$ and $\beta_x$ functions (see Eq. 3), it is convenient to reformulate the governing equation in the form of Eq. 4, yielding for

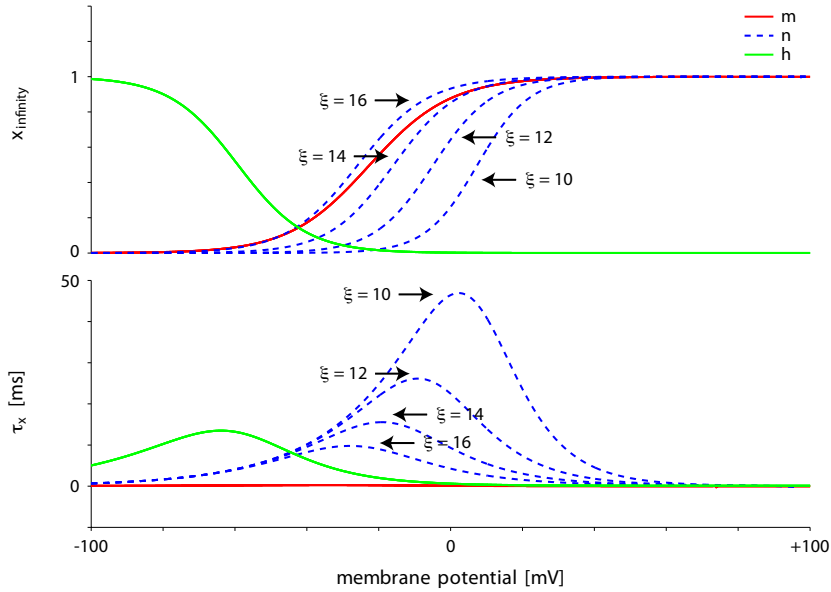

Figure 3: Equilibrium function $x_\infty$ (top) and time constant $\tau_x$ (bottom) as a function of the membrane voltage for different values of $\xi$ for the gating variables $m$ (red line), $h$ (green line) and $n$ (dotted blue lines).

each gating variable an equilibrium value $x_\infty(V)$ and a time constant $\tau_x(V)$. Figure 3 shows $x_\infty$ and $\tau_x$ for all three gating variables of the model as a function of the membrane voltage $V$, the variable opening the sodium channel $m$, the variable closing the sodium channel $h$ and the variable associated with the potassium channel $n$. With increment in the value of $\xi$, the asymptotic value $n_\infty$ shifts towards lower membrane potentials, in other words for the same membrane voltage, the equilibrium value is higher. On the opposite, with increment in the value of $\xi$, the time constant $\tau_x$ is reduced in the range $[-40; +40]$ mV. In summary, at low $\xi$ values, the potassium current iK is only activated when the membrane potential is high and it kicks in slowly. At high $\xi$ values, iK is activated earlier in the action potential and kicks in faster. This supports remarkably well the arguments of Carter and Bean to explain the relative metabolic inefficiency of GABAergic neurons. Indeed, fast-spiking neurons with narrow action potentials use fast-activating Kv3 channels to repolarize the membrane. It is postulated that, in these cells, recovery begins sooner and from more hyperpolarized voltages in remarkable agreement with the evolution of $n_\infty$ and $\tau_n$ in our modeling framework. It is also interesting to note that Kv3 channels enable fast spiking [8]. This is supposedly due to incomplete sodium channel inactivation and to earlier recovery, in effect speeding recovery and reducing the refractory period.

Finally, Figure 4 shows the membrane voltage $V$ when the model is subjected to a constant input as well as the corresponding gain functions or frequency versus current curves. The $f - I$ curve has the typical saturating profile observed for many neurons [9] and all the models start spiking at a non-zero frequency. In line with the idea that neurons with a sharp action potential and incomplete inactivation of sodium channels can spike faster, the discharge frequency increases with the value of $\xi$ for a given input current.

## 4  Discussion

Recent experimental results have highlighted that the original Hodgkin-Huxley model [1] is not particularly well suited to describe the dynamics of sodium and potassium voltage-gated channels during the course of an action potential in mammalian neurons. The Hodgkin-Huxley model is also a poor foundation for studies dedicated to computing an energy budget for the mammalian brain since it severely overestimates the metabolic cost associated with action potentials by at least a factor of 2. Despite that, the Hodgkin-Huxley model is still widely used and often for modeling projects specifically targeting the mammalian brain.

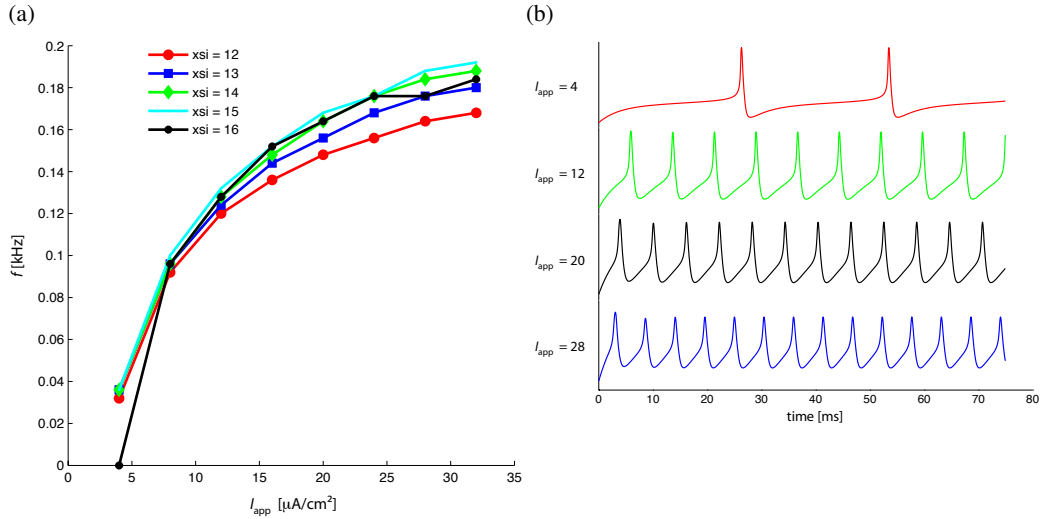

Figure 4: Gain functions and spike trains elicited by constant input. (a) The gain function ($f − I$ curve) is plotted for different values of the parameter $\xi$. The models were stimulated with a constant current input of 5 sec after an initial 30 ms pulse. (b) Sample spike trains for $\xi = 14$ for different values of the externally applied current $I_{\text{ext}}$.

Here we have introduced a novel instance of the Hodgkin-Huxley model aimed at correcting these issues. The proposed family of models uses the original equations of Hodgkin and Huxley as they were formulated originally but introduces new expressions for the functions $\alpha_x$ and $\beta_x$ that characterize the dynamics of the gating variables $m$, $n$ and $h$. Moreover, the specific expression for $\alpha_n$ depends on an extra parameter $\xi$. By varying $\xi$ in a specific range, our family of models is able to quantitatively reproduce a wide range of dynamics for the voltage-gated sodium and potassium channels during individual action potentials. Our family of models is able to generate broad, metabolically efficient action potentials with a sharp single peak dynamics of the sodium current as well as narrow, metabolically inefficient action potentials with incomplete inactivation of the sodium channels during the decaying phase of the action potential. These different behaviors cover neuron types as different as cortical pyramidal neurons, cortical interneurons or Purkinje cells.

For this study we chose a single-compartment Hodgkin-Huxley-type model because it is well suited to compare with the experimental conditions of Carter and Bean [6]. However, when comparing the particular parameterization of the model that is achieved here and experimental data (see Figure 2), it suggests that other changes, e.g. in sodium channel inactivation, may help to explain the differences between different cell types. It should also be noted that action potentials as narrow as 250 $\mu$s can be as energy-efficient (SER = 1.3) [10] as the widest action potentials measured by Carter and Bean [6], suggesting that sodium channel kinetics, in addition to potassium channel kinetics, is also different for different cell types and subcellular compartments.

Numerous studies have been dedicated to study the energy constraints of the brain from the coding and network design perspective [4,11] or from the channel kinetic perspective [3,5,6,12]. Recently it has been argued that energy minimization under functional constraints could be the unifying principle governing the specific combination of ion channels that each individual neuron expresses [12]. In support of this hypothesis, it was demonstrated that some mammalian neurons generate their action potentials with currents that almost reach optimal metabolic efficiency [5]. So far, these studies have mostly addressed the question of metabolic efficiency considering isolated action potentials. Moreover, it can be difficult to compare neurons with very different properties. Here, we have introduced a new family of biophysical models able to reproduce different action potentials relevant to this debate and their underlying currents [6]. We believe that our approach is very valuable in providing mechanistic insights into the specific properties of different types of neurons. It also suggests that it could be possible to design a generic Hodgkin-Huxley-type model family that could encompass a very broad range of different observed behaviors in a similar way than the Izhikevich model does

for integrate-and-fire type model neurons [13]. Finally we believe that our model family will prove invaluable in studying metabolic questions and in particular in addressing the specific question: why are inhibitory neurons less metabolically efficient than excitatory neurons?

## Acknowledgements

RJ is supported by grants from the Olga Mayenfisch Foundation and from the Hartmann Müller Foundation. The authors would like to thank Dr Arnd Roth for helpful discussions.

## References

[1] Hodgkin AL, Huxley AF. J Physiol 1952; 116: 449–472.

[2] Destexhe A, Paré D. J Neurophysiol 1999; 81: 1531–1547.

[3] Sengupta B, Stemmler M, Laughlin SB, Niven JE. PLoS Comp. Biol. 2010; 6: e1000840.

[4] Attwell D, Laughlin SB. J Cereb Blood Flow Metab 2001; 21: 1133–1145.

[5] Alle H, Roth A, Geiger J. Science 2009; 325: 1405–1408.

[6] Carter BC, Bean BP. Neuron 2009; 64: 898–909.

[7] Jolivet R, Lewis TJ, Gerstner W. J Neurophysiol 2004; 92: 959–976.

[8] Lien CC, Jonas P. J Neurosci 2003; 23: 2058–2068.

[9] Rauch A, La Camera G, Lüscher HR, Senn W, Fusi S. J Neurophysiol 2003; 90: 1598–1612.

[10] Alle H and Geiger J. Science 2006; 311: 1290–1293.

[11] Laughlin SB, Sejnowski T. Science 2003; 301: 1870–1874.

[12] Hasenstaub A, Otte S, Callaway E, Sejnowski TJ. PNAS 2010; 107: 12329–12334.

[13] Izhikevich E. IEEE Trans Neural Net 2003; 14: 1569- 1572.

